# COMPARING BIASES FOR MINIMAL NETWORK CONSTRUCTION WITH BACK-PROPAGATION

Stephen José Hanson†
Bell Communications Research
Morristown, New Jersey 07960

Lorien Y. Pratt
Rutgers University
New Brunswick, New Jersey 08903

## ABSTRACT

Rumelhart (1987), has proposed a method for choosing minimal or "simple" representations during learning in Back-propagation networks. This approach can be used to (a) dynamically select the number of hidden units, (b) construct a representation that is appropriate for the problem and (c) thus improve the generalization ability of Back-propagation networks. The method Rumelhart suggests involves adding penalty terms to the usual error function. In this paper we introduce Rumelhart's minimal networks idea and compare two possible biases on the weight search space. These biases are compared in both simple counting problems and a speech recognition problem. In general, the constrained search does seem to minimize the number of hidden units required with an expected increase in local minima.

## INTRODUCTION

Many supervised connectionist models use gradient descent in error to solve various kinds of tasks (Rumelhart, Hinton & Williams, 1986). However, such gradient descent methods tend to be "opportunistic" and can solve problems in an arbitrary way dependent on starting point in weight space and peculiarities of the training set. For example, in Figure 1 we show a "mesh" problem which consists of a random distribution of exemplars from two categories. The spatial geometry of the categories impose a meshed or overlapping subset of the exemplars in the two dimensional feature space. As the meshed part of the categories increase the problem becomes more complex and must involve the combination of more linear cuts in feature space and consequently more nonlinear cuts for category separation. In the top left corner of Figure 1(a), we show a mesh geometry requiring only three cuts for category separation. In the bottom center

1(b) is the projection of the three cut solution of the mesh in output space. In the top right of this Figure 1(c) is a typical solution provided by back-propagation starting with 16 hidden units. This Figure shows the two dimensional feature space in which 9 of the linear cuts are projected (the other 7 are outside the [0,1] unit plane).

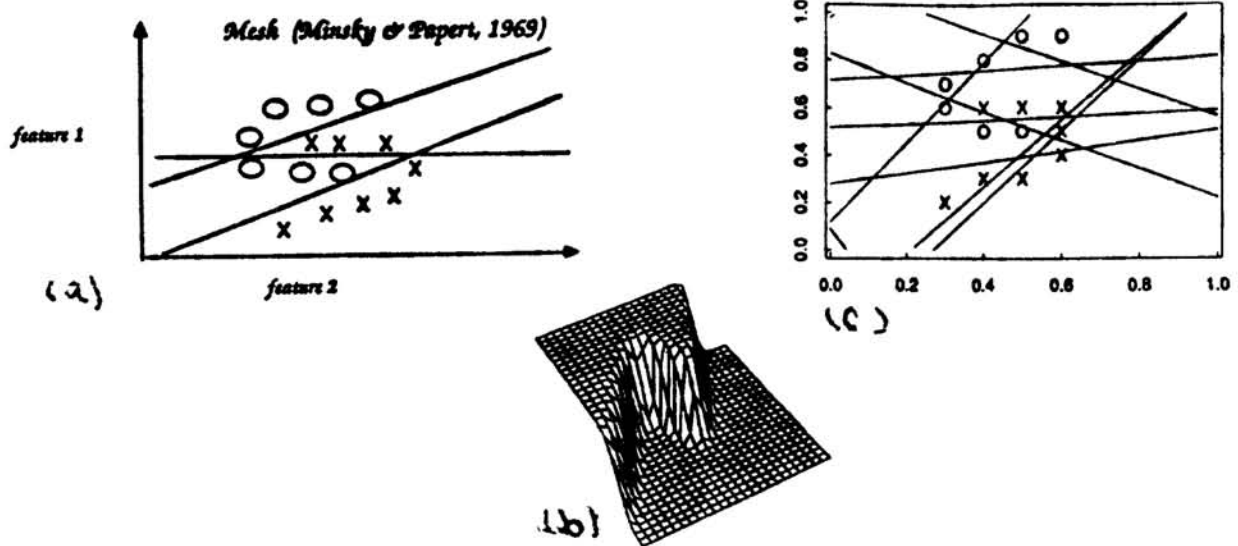

Figure 1: Mesh problem (a), output space (b) and typical back-propagation solution (c)

Examining the weights in the next layer of the network indicates that in fact, 7 of these 9 line segments are used in order to construct the output surface shown in Figure 1(b). Consequently, the underlying feature relations determining the output surface and category separation are arbitrary, more complex then necessary and may result in anomalous generalizations.

Rumelhart (1987), has proposed a way to increase the generalization capabilities of learning networks which use gradient descent methods and to automatically control the resources learning networks use-for example, in terms of "hidden" units. His hypothesis concerns the nature of the representation in the network: "...*the simplest most robust network which accounts for a data set will, on average, lead to the best generalization to the population from which the training set has been drawn*".

The basic approach involves adding penalty terms to the usual error function in order to constrain the search and cause weights to differentially decay. This is similar to many proposals in statistical regression where a "simplicity" measure is minimized along with the error term and is sometimes referred to as "biased" regression (Rawlings, 1988). Basically, the statistical concept of biased regression derives from parameter estimation approaches that attempt to achieve a best linear unbiased estimator ("BLUE"). By definition an unbiased estimator is one with the lowest possible variance and theoretically, unless there is significant collinearity[1] or nonlinearity amongst the

---

1. For example, *Ridge regression* is a special case of biased regression which attempts to make a singular correlation matrix non-singular by adding a small arbitrary constant to the diagonal of the matrix. This increase in the diagonal may lower the impact of the off-diagonal elements and thus reduce the effects of collinearity.

variables, a least squares estimator (LSE) can be also shown to be a BLUE. If on the other hand, input variables are correlated or nonlinear with the output variables (as is the case in back-propagation) then there is no guarantee that the LSE will also be unbiased. Consequently, introducing a bias may actually reduce the variance of the estimator of that below the theoretically unbiased estimator.

Since back-propagation is a special case of multivariate nonlinear regression methods we must immediately give up on achieving a BLUE. Worse yet, the input variables are also very likely to be collinear in that input data are typically expected to be used for feature extraction. Consequently, the neural network framework leads naturally to the exploration of biased regression techniques, unfortunately, it is not obvious what sorts of biases ought to be introduced and whether they may be problem dependent. Furthermore, the choice of particular biases probably determines the particular representation that is chosen and its nature in terms of size, structure and "simplicity". This representation bias may in turn induce generalization behavior which is greater in accuracy with larger coverage over the domain. Nonetheless, since there is no particular motivation for minimizing a least squares estimator it is important to begin exploring possible biases that would lead to lower variance and more robust estimators.

In this paper we explore two general type of bias which introduce explicit constraints on the hidden units. First we discuss the standard back-propagation method, various past methods of biasing which have been called "weight decay", the properties of our biases, and finally some simple benchmark tests using parity and a speech recognition task.

## BACK-PROPAGATION

The Back-propagation method [2] is a supervised learning technique using a gradient descent in an error variable. The error is established by comparing an output value to a desired or expected value. These errors can be accumulated over the sample:

$$E = \sum_s \sum_i (y_{is} - \hat{y}_{is})^2 \qquad (1)$$

Assuming the output function is differentiable then a gradient of the error can be found, and we require that this derivative be decreasing,

$$-\frac{\partial E}{\partial w_{ij}} = 0. \qquad (2)$$

Over multiple layers we pass back a weighted sum of each derivative from units above.

## WEIGHT DECAY

Past work[2] using biases have generally been based on ad hoc arguments that weights should differentially decay allowing large weights to persist and small weights to tend

towards zero sooner. Apparently, this would tend to concentrate more of the input into a smaller number of weights. Generally, the intuitive notion is to somehow reduce the complexity of the network as defined by the number of connections and number of hidden units. A simple but inefficient way of doing this is to include a weight decay term in the usual delta updating rule causing all weights to decay on each learning step (where $w = w_{ij}$ *throughout*):

$$w_{n+1} = \alpha \left(-\frac{\partial E}{\partial w}\right)_n + \beta \, w_n \qquad (3)$$

Solving this difference equation shows that for $\beta < 1.0$ weights are decaying exponentially over steps towards zero,

$$w_n = \alpha \sum_{i=1}^{n} \beta^{n-i} \left(-\frac{\partial E}{\partial w}\right)_i + \beta^n \, w_0 \qquad (4)$$

This approach introduces the decay term in the derivative itself causing error terms to also decrease over learning steps which may not be desirable.

## BIAS

The sort of weight decay just discussed can also be derived from general consideration of "costs" on weights. For example it is possible to consider E with a bias term which in the simple decay case is quadratic with weight value (i.e. $w^2$).

We now combine this bias with E producing an objective function that includes both the error term and this bias function:

$$O = E + B \qquad (5)$$

where, we now want to minimize

$$\frac{\partial O}{\partial w_{ij}} = \frac{\partial E}{\partial w_{ij}} + \frac{\partial B}{\partial w_{ij}} \qquad (6)$$

In the quadratic case the updating rule becomes,

$$w_{n+1} = \alpha \left(-\frac{\partial E}{\partial w_{ij}} - 2w_n\right) + w_n \qquad (7)$$

Solving this difference equation derives the updating rule from equation 4.

$$w_n = \alpha \sum_{i=1}^{n} (1-2\alpha)^{n-i} \left(-\frac{\partial E}{\partial w_{ij}}\right)_i + (1-2\alpha)^n w_0 \qquad (8)$$

In this case, however without introduction of other parameters, $\alpha$ is both the learning rate

---

2.  Most of the work discussed here has not been previously published but nonetheless has entered into general use in many connectionist models and was recently summarized on the *Connectionist Bulletin Board* by John Kruschke.

and related to the decay term and must be strictly $< \frac{1}{2}$ for weight decay.

Uniform weight decay has a disadvantage in that large weights are decaying at the same rate as small weights. It is possible to design biases that influence weights only when they are relatively small or even in a particular range of values. For example, Rumelhart has entertained a number of biases, one form in particular that we will also explore is based on a rectangular hyperbolic function,

$$B = \frac{w^2}{(1+w^2)} \qquad (9)$$

It is informative to examine the derivative associated with this function in order to understand its effect on the weight updates.

$$-\frac{\partial B}{\partial w_{ij}} = -\frac{2w}{(1+w^2)^2} \qquad (10)$$

This derivative is plotted in Figure 2 (indicated as Rumelhart) and is non-monotonic showing a strong differential effect on small weights (+ or -) pushing them towards zero, while near zero and large weight values are not significantly affected.

## BIAS PER UNIT

It is possible to consider bias on each hidden unit weight group. This has the potentially desirable effect of isolating weight changes to hidden unit weight groups and could effectively eliminate hidden units. Consequently, the hidden units are directly determining the bias. In order to do this, first define

$$w_i = \sum_j |w_{ij}|, \qquad (11)$$

where i is the ith hidden unit.

### Hyperbolic Bias

Now consider a function similar to Rumelhart's but this time with $w_i$, the ith hidden group as the variable,

$$B = \frac{w_i}{1 + \lambda\, w_i}. \qquad (12)$$

The new gradient includes the term from the bias which is,

$$-\frac{\partial B}{\partial w_{ij}} = -\frac{\lambda sgn\,(w_{ij})}{(1+w_i)^2} \qquad (13)$$

### Exponential Bias

A similar kind of bias would be to consider the negative exponential:

$$B = (1-e^{-\lambda w_i}) \tag{14}$$

This bias is similar to the hyperbolic bias term as above but involves the exponential which potentially produce more uniform and gradual rate changes towards zero,

$$-\frac{\partial B}{\partial w_{ij}} = -\frac{sgn(w_{ij})}{(e^{\lambda w_i})}. \tag{15}$$

The behavior of these two biases (hyperbolic, exponential) are shown as function of weight magnitudes in Figure 2. Notice that the exponential bias term is more similar in slope change to Rumelhart's (even though his is non-monotonic) than the hyperbolic as weight magnitude to a hidden unit increases.

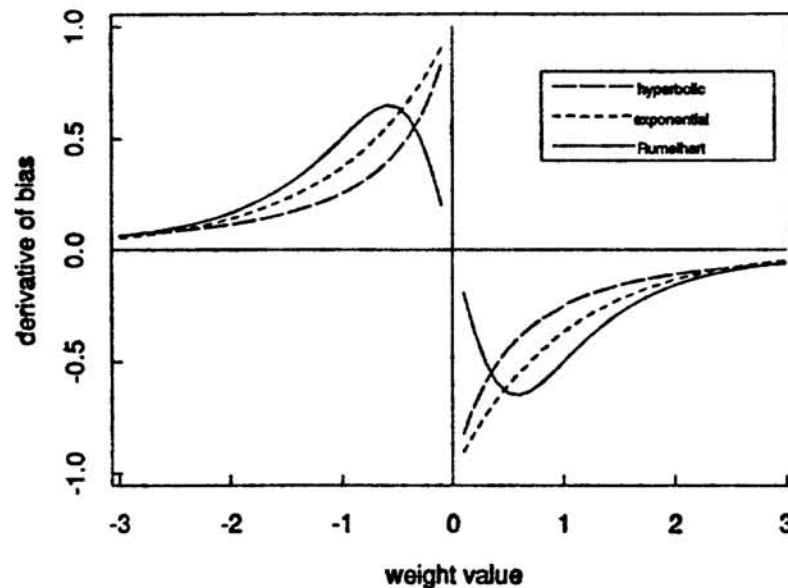

Figure 2: Bias function behavior of Rumelhart's, Hyperbolic and Exponential

Obviously there are many more kinds of bias that one can consider. These two were chosen in order to provide a systematic test of varying biases and exploring their differential effectiveness in minimizing network complexity.

## SOME COMPARISONS

### Parity

These biased Back-propagation methods were applied to several counting problems and to a speech (digit) recognition problem. In the following graphs for example, we show the results of 100 runs of XOR and 4-bit parity at $\eta = .1$ (learning rate) and $\alpha = .8$ (moving average) starting with 10 hidden units. The parameter $\lambda$ was optimized for the bias runs.

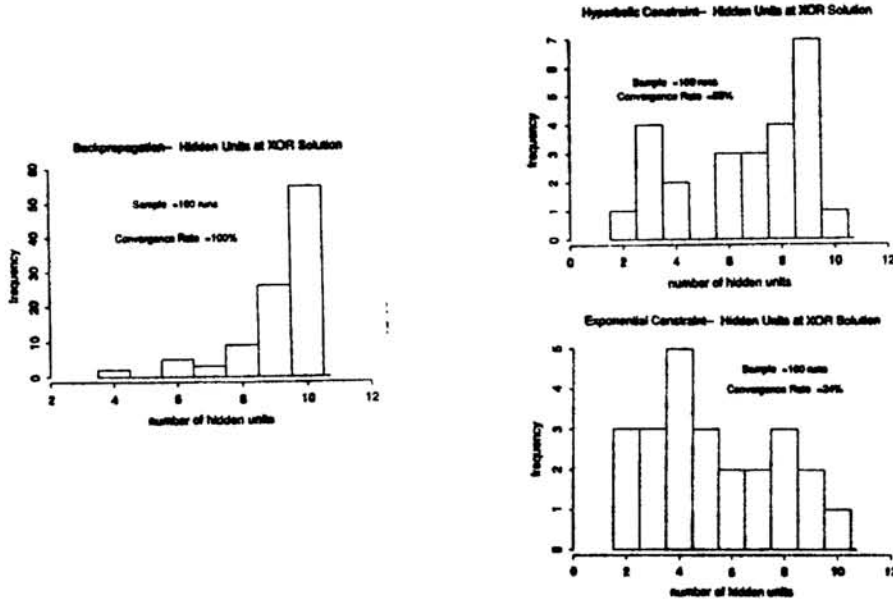

Figure 3: Exclusive OR runs for standard, hyperbolic and exponential biasing

Shown are runs for the standard case without biases, the hyperbolic bias and the exponential bias. Once a solution was reached all hidden units were tested individually by removing each of them one at a time from the network and then testing on the training set. Any hidden unit which was unnecessary was removed for data analysis. Only the number of these "functional units" are reported in the histograms. Notice the number of hidden units decrease with bias runs. An analysis of variance (statistical test) verified this improvement for both the hyperbolic and exponential over the standard. Also note that the exponential is significantly better than the hyperbolic. This is also confirmed for the 4-bit parity case as shown in Figure 4.

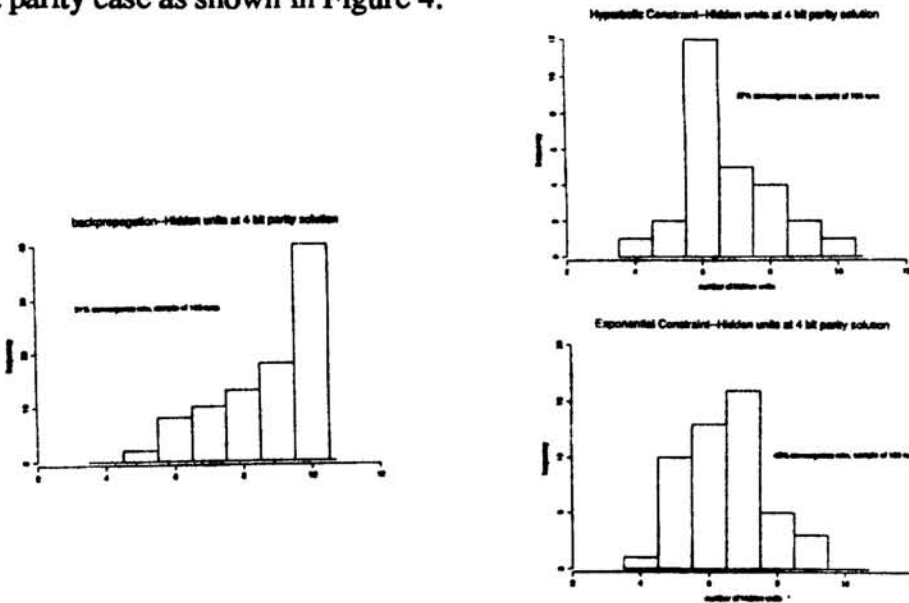

Figure 4: four-bit parity runs for standard, hyperbolic and exponential biasing

## Speech Recognition

Samples of 10 spoken digits (0-9) each were collected (same speaker throughout--D.J. Burr kindly supplied data). Samples were then preprocessed using FFTs retaining the first 12 Cepstral coefficients. To avoid ceiling effects only two tokens each of the 10 digits were used for training ("0", "0","1","1",....."9","9"..) each network. Eight such 2 token samples were used for replications. Another set of 50 spoken digits (5 samples of each of the 10 digits) were collected for transfer. All runs were matched across methods for number of learning sweeps (<300), η=.05, α=.2, and λ=.01 which were optimized for the exponential bias. Shown in the following table is the results of the 8 replications for the standard and the exponential bias.

| Sample | backpropagation | | constrained (exp) | |
|---|---|---|---|---|
| | Transfer | # Hidden Units | Transfer | # Hidden Units |
| r1 | 50% | 18 | 64% | 10 |
| r2 | 60% | 17 | 76% | 13 |
| r3 | 62% | 18 | 64% | 14 |
| r4 | 66% | 14 | 74% | 14 |
| r5 | 62% | 16 | 56% | 11 |
| r6 | 66% | 19 | 68% | 14 |
| r7 | 58% | 18 | 54% | 11 |
| r8 | 58% | 18 | 64% | 9 |

$$\bar{x} \pm \frac{s}{\sqrt{n}} \qquad 59\%\pm1.9 \qquad 17\pm.56 \qquad 65\%\pm2.8 \qquad 12\pm.71$$

Table 1:  Eight replications with transfer for standard and exponential bias.

In this case there appears to both an improvement in the average number of hidden units (functional ones) and transfer. A typical correlation of the improved transfer and reduced hidden unit usage for a single replication is plotted in the next graph.

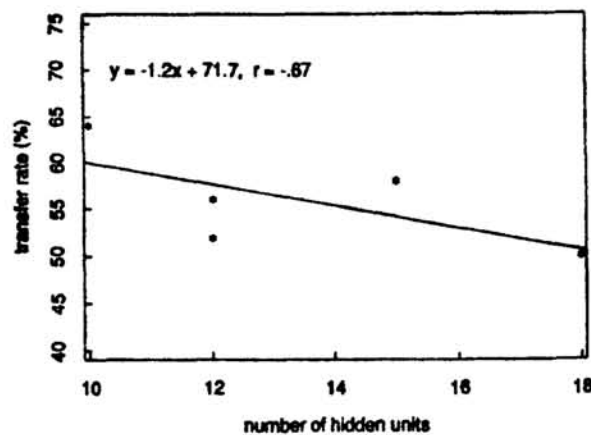

Figure 5:  Transfer as a function of hidden unit usage for a single replication

We note that introduction of biases decrease the probability of convergence relative to the standard case (as many as 75% of the parity runs did not converge within criteria

number of sweeps.) Since the search problem is made more difficult by introducing biases it now becomes even more important to explore methods for improving convergence similar for example, to simulated annealing (Kirkpatrick, Gelatt & Vecchi, 1983)

# CONCLUSIONS

Minimal networks were defined and two types of bias were compared in a simple counting problem and a speech recognition problem. In the counting problems under biasing conditions the number hidden units tended to decrease towards the minimum required for the problem although with a concomitant decrease in convergence rate. In the speech problem also under biasing conditions the number of hidden units tended to decrease as the transfer rate tended to improve.

### Acknowledgements

We thank Dave Rumelhart for discussions concerning the minimal network concept, the Bellcore connectionist group and members of the Princeton Cognitive Science Lab for a lively environment for the development of these ideas.

## Footnotes

† Also member of Cognitive Science Laboratory, 221 Nassau Street, Princeton University, Princeton, New Jersey, 08542

### References

Kirkpatrick, S., Gelatt, C. D., & Vecchi, M. P., Optimization by simulated annealing. Science, 220, 671-680, (1983).

Rawlings, J. O., Applied Regression Analysis, Wadsworth & Brooks/Cole, (1988).

Rumelhart D. E., Personal Communication, Princeton, (1987).

Rumelhart D. E., Hinton G. E., & Williams R., Learning Internal Representations by error propagation. Nature, (1986).
